# Reducing Spike Train Variability:
# A Computational Theory Of
# Spike-Timing Dependent Plasticity

**Sander M. Bohte**[1,2]
*S.M.Bohte@cwi.nl*
[1]Dept. Software Engineering
CWI, Amsterdam, The Netherlands

**Michael C. Mozer**[2]
*mozer@cs.colorado.edu*
[2]Dept. of Computer Science
University of Colorado, Boulder, USA

## Abstract

Experimental studies have observed synaptic potentiation when a presynaptic neuron fires shortly before a postsynaptic neuron, and synaptic depression when the presynaptic neuron fires shortly after. The dependence of synaptic modulation on the precise timing of the two action potentials is known as *spike-timing dependent plasticity* or *STDP*. We derive STDP from a simple computational principle: synapses adapt so as to minimize the postsynaptic neuron's variability to a given presynaptic input, causing the neuron's output to become more reliable in the face of noise. Using an entropy-minimization objective function and the biophysically realistic spike-response model of Gerstner (2001), we simulate neurophysiological experiments and obtain the characteristic STDP curve along with other phenomena including the reduction in synaptic plasticity as synaptic efficacy increases. We compare our account to other efforts to derive STDP from computational principles, and argue that our account provides the most comprehensive coverage of the phenomena. Thus, reliability of neural response in the face of noise may be a key goal of cortical adaptation.

## 1  Introduction

Experimental studies have observed synaptic potentiation when a presynaptic neuron fires shortly before a postsynaptic neuron, and synaptic depression when the presynaptic neuron fires shortly after. The dependence of synaptic modulation on the precise timing of the two action potentials, known as *spike-timing dependent plasticity* or *STDP*, is depicted in Figure 1. Typically, plasticity is observed only when the presynaptic and postsynaptic spikes (hereafter, *pre* and *post*) occur within a 20–30 ms time window, and the transition from potentiation to depression is very rapid. Another important observation is that synaptic plasticity decreases with increased synaptic efficacy. The effects are long lasting, and are therefore referred to as long-term potentiation (LTP) and depression (LTD). For detailed reviews of the evidence for STDP, see [1, 2].

Because these intriguing findings appear to describe a fundamental learning mechanism in the brain, a flurry of models have been developed that focus on different aspects of STDP, from biochemical models that explain the underlying mechanisms giving rise to STDP [3], to models that explore the consequences of a STDP-like learning rules in an ensemble of spiking neurons [4, 5, 6, 7], to models that propose fundamental computational justifications for STDP. Most commonly, STDP

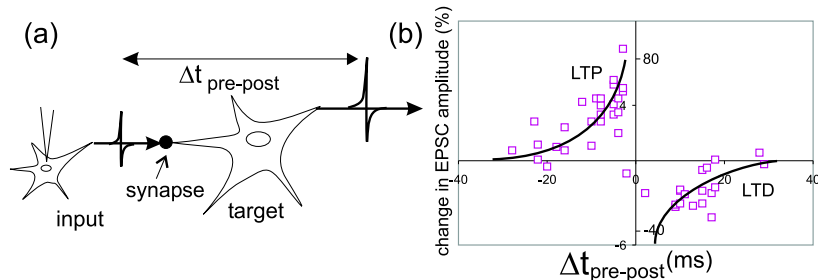

Figure 1: (a) Measuring STDP experimentally: pre-post spike pairs are repeatedly induced at a fixed interval $\Delta t_{pre-post}$, and the resulting change to the strength of the synapse is assessed; (b) change in synaptic strength after repeated spike pairing as a function of the difference in time between the pre and post spikes (data from Zhang *et al.*, 1998). We have superimposed an exponential fit of LTP and LTD.

is viewed as a type of asymmetric Hebbian learning with a temporal dimension. However, this perspective is hardly a fundamental computational rationale, and one would hope that such an intuitively sensible learning rule would emerge from a first-principle computational justification.

Several researchers have tried to derive a learning rule yielding STDP from first principles. Rao and Sejnowski [8] show that STDP emerges when a neuron attempts to predict its membrane potential at some time $t$ from the potential at time $t - \Delta t$. However, STDP emerges only for a narrow range of $\Delta t$ values, and the qualitative nature of the modeling makes it unclear whether a quantitative fit can be obtained. Dayan and Häusser [9] show that STDP can be viewed as an optimal noise-removal filter for certain noise distributions. However, even small variation from these noise distributions yield quite different learning rules, and the noise statistics of biological neurons are unknown. Eisele (private communication) has shown that an STDP-like learning rule can be derived from the goal of maintaining the relevant connections in a network. Chechik [10] is most closely related to the present work. He relates STDP to information theory via maximization of mutual information between input and output spike trains. This approach derives the LTP portion of STDP, but fails to yield the LTD portion.

The computational approach of Chechik (as well as Dayan and Häusser) is premised on a rate-coding neuron model that disregards the relative timing of spikes. It seems quite odd to argue for STDP using rate codes: if spike timing is irrelevant to information transmission, then STDP is likely an artifact and is not central to understanding mechanisms of neural computation. Further, as noted in [9], because STDP is not quite additive in the case of multiple input or output spikes that are near in time [11], one should consider interpretations that are based on individual spikes, not aggregates over spike trains.

Here, we present an alternative computational motivation for STDP. We conjecture that a fundamental objective of cortical computation is to achieve *reliable* neural responses, that is, neurons should produce the identical response—both in the number and timing of spikes—given a fixed input spike train. Reliability is an issue if neurons are affected by noise influences, because noise leads to variability in a neuron's dynamics and therefore in its response. Minimizing this variability will reduce the effect of noise and will therefore increase the informativeness of the neuron's output signal. The source of the noise is not important; it could be intrinsic to a neuron (e.g., a noisy threshold) or it could originate in unmodeled external sources causing fluctuations in the membrane potential uncorrelated with a particular input.

We are not suggesting that increasing neural reliability is the *only* learning objective.

If it were, a neuron would do well to give no response regardless of the input. Rather, reliability is but one of many objectives that learning tries to achieve. This form of unsupervised learning must, of course, be complemented by supervised and reinforcement learning that allow an organism to achieve its goals and satisfy drives.

We derive STDP from the following computational principle: synapses adapt so as to minimize the entropy of the postsynaptic neuron's output in response to a given presynaptic input. In our simulations, we follow the methodology of neurophysiological experiments. This approach leads to a detailed fit to key experimental results. We model not only the shape (sign and time course) of the STDP curve, but also the fact that potentiation of a synapse depends on the efficacy of the synapse—it decreases with increased efficacy. In addition to fitting these key STDP phenomena, the model allows us to make predictions regarding the relationship between properties of the neuron and the shape of the STDP curve.

Before delving into the details of our approach, we attempt to give a basic intuition about the approach. Noise in spiking neuron dynamics leads to variability in the number and timing of spikes. Given a particular input, one spike train might be more likely than others, but the output is nondeterministic. By the entropy-minimization principle, adaptation should reduce the likelihood of these other possibilities. To be concrete, consider a particular experimental paradigm. In [12], a pre neuron is identified with a weak synapse to a post neuron, such that the pre is unlikely to cause the post to fire. However, the post can be induced to fire via a second presynaptic connection. In a typical trial, the pre is induced to fire a single spike, and with a variable delay, the post is also induced to fire (typically) a single spike. To increase the likelihood of the observed post response, other response possibilities must be suppressed. With presynaptic input preceding the postsynaptic spike, the most likely alternative response is no output spikes at all. Increasing the synaptic connection weight should then reduce the possibility of this alternative response. With presynaptic input following the postsynaptic spike, the most likely alternative response is a second output spike. Decreasing the synaptic connection weight should reduce the possibility of this alternative response. Because both of these alternatives become less likely as the lag between pre and post spikes is increased, one would expect that the magnitude of synaptic plasticity diminishes with the lag, as is observed in the STDP curve.

Our approach to reducing response variability given a particular input pattern involves computing the gradient of synaptic weights with respect to a differentiable model of spiking neuron behavior. We use the Spike Response Model (SRM) of [13] with a stochastic threshold, where the stochastic threshold models fluctuations of the membrane potential or the threshold outside of experimental control. For the stochastic SRM, the response probability is differentiable with respect to the synaptic weights, allowing us to calculate the entropy gradient with respect to the weights conditional on the presented input. Learning is presumed to take a gradient step to reduce this conditional entropy. In modeling neurophysiological experiments, we demonstrate that this learning rule yields the typical STDP curve. We can predict the relationship between the exact shape of the STDP curve and physiologically measurable parameters, and we show that our results are robust to the choice of the few free parameters of the model.

Two papers in these proceedings are closely related to our work. They also find STDP-like curves when attempting to maximize an information-theoretic measure—the mutual information between input and output—for a Spike Response Model [14, 15]. Bell & Parra [14] use a deterministic SRM model which does not model the LTD component of STDP properly. The derivation by Toyoizumi et al. [15] is valid only for an essentially constant membrane potential with small fluctuations. Neither of these approaches has succeeded in quantitatively modeling specific experimental

data with neurobiologically-realistic timing parameters, and neither explains the saturation of LTD/LTP with increasing weights as we do. Nonetheless, these models make an interesting contrast to ours by suggesting a computational principle of optimization of information transmission, as contrasted with our principle of neural noise reduction. Perhaps experimental tests can be devised to distinguish between these competing theories.

## 2 The Stochastic Spike Response Model

The Spike Response Model (SRM), defined by Gerstner [13], is a generic integrate-and-fire model of a spiking neuron that closely corresponds to the behavior of a biological spiking neuron and is characterized in terms of a small set of easily interpretable parameters [16]. The standard SRM formulation describes the temporal evolution of the membrane potential based on past neuronal events, specifically as a weighted sum of postsynaptic potentials (PSPs) modulated by reset and threshold effects of previous postsynaptic spiking events. Following [13], the membrane potential of cell $i$ at time $t$, $u_i(t)$, is defined as:

$$u_i(t) = \eta(t - \hat{f}_i) + \sum_{j \in \Gamma_i} w_{ij} \sum_{f_j \in \mathcal{F}_j^t} \varepsilon(t - \hat{f}_i, t - f_j), \tag{1}$$

where $\Gamma_i$ is the set of inputs connected to neuron $i$, $\mathcal{F}_j^t$ is the set of times prior to $t$ that neuron $j$ has spiked, $\hat{f}_i$ is the time of the last spike of neuron $i$, $w_{ij}$ is the synaptic weight from neuron $j$ to neuron $i$, $\varepsilon(t - \hat{f}_i, t - f_j)$ is the PSP in neuron $i$ due to an input spike from neuron $j$ at time $f_j$, and $\eta(t - \hat{f}_i)$ is the refractory response due to the postsynaptic spike at time $\hat{f}_i$. Neuron $i$ fires when the potential $u_i(t)$ exceeds a threshold ($\vartheta$) from below.

The postsynaptic potential $\varepsilon$ is modeled as the differential alpha function in [13], defined with respect to two variables: the time since the most recent postsynaptic spike, $x$, and the time since the presynaptic spike, $s$:

$$\varepsilon(x, s) = \frac{1}{1 - \frac{\tau_s}{\tau_m}} \Big\{ \big[ \exp\big(-\frac{s}{\tau_m}\big) - \exp\big(-\frac{s}{\tau_s}\big) \big] \mathcal{H}(s) \mathcal{H}(x - s) + \tag{2}$$
$$+ \exp\big(-\frac{s - x}{\tau_s}\big) \big[ \exp\big(-\frac{x}{\tau_m}\big) - \exp\big(-\frac{x}{\tau_s}\big) \big] \mathcal{H}(x) \mathcal{H}(s - x) \Big\},$$

where $\tau_s$ and $\tau_m$ are the rise and decay time-constants of the PSP, and $\mathcal{H}$ is the Heaviside function. The refractory reset function is defined to be [13]:

$$\eta(x) = u_{abs} \mathcal{H}(\Delta_{abs} - x) \mathcal{H}(-x) + u_{abs} \exp\big(-\frac{x + \Delta_{abs}}{\tau_r^f}\big) + u_r^s \exp\big(-\frac{x}{\tau_r^s}\big), \tag{3}$$

where $u_{abs}$ is a large negative contribution to the potential to model the absolute refractory period, with duration $\Delta_{abs}$. We smooth this refractory response by a fast decaying exponential with time constant $\tau_r^f$. The third term in the sum represents the slow decaying exponential recovery of an elevated threshold, $u_r^s$, with time constant $\tau_r^s$. (Graphs of these $\varepsilon$ and $\eta$ functions can be found in [13].) We made a minor modification to the SRM described in [13] by relaxing the constraint that $\tau_r^s = \tau_m$; smoothing the absolute refractory function is mentioned in [13] but not explicitly defined as we do here. In all simulations presented, $\Delta_{abs} = 2ms$, $\tau_r^s = 4\tau_m$, and $\tau_r^f = 0.1\tau_m$.

The SRM we just described is deterministic. Gerstner [13] introduces a stochastic variant of the SRM (*sSRM*) by incorporating the notion of a stochastic firing threshold: given membrane potential $u_i(t)$, the probability density of the neuron firing at time $t$ is specified by $\rho(u_i(t))$. Herrmann & Gerstner [17] find that then for a realistic escape-rate noise model the firing probability density as a function of the potential is initially small and constant, transitioning to asymptotically linear

increasing around threshold $\vartheta$. In our simulations, we use such a function:

$$\rho(v) = \frac{\beta}{\alpha}(\ln[1 + \exp(\alpha(\vartheta - v))] - \alpha(\vartheta - v)), \quad (4)$$

where $\vartheta$ is the firing threshold in the absence of noise, $\alpha$ determines the abruptness of the constant-to-linear probability density transition around $\vartheta$, and $\beta$ determines the slope of the increasing part. Experiments with sigmoidal and exponential density functions were found to not qualitatively affect the results.

## 3   Minimizing Conditional Entropy

We now derive the rule for adjusting the weight from a presynaptic neuron $j$ to a postsynaptic sSRM neuron $i$, so as to minimize the entropy of $i$'s response given a particular spike sequence from $j$. A spike sequence is described by the set of all times at which spikes have occurred within some interval between 0 and $T$, denoted $\mathcal{F}_j^T$ for neuron $j$. We assume the interval is wide enough that spikes outside the interval do not influence the state of the neuron within the interval (e.g., through threshold reset effects). We can then treat intervals as independent of each other.

Let the postsynaptic neuron $i$ produce a response $\xi \in \Omega_i$, where $\Omega_i$ is the set of all possible responses given the input, $\xi \equiv \mathcal{F}_i^T$, and $g(\xi)$ is the probability density over responses. The differential conditional entropy $h(\Omega_i)$ of neuron $i$'s response is then defined as:

$$h(\Omega_i) = -\int_{\Omega_i} g(\xi)\log\big(g(\xi)\big)d\xi. \quad (5)$$

To minimize the differential conditional entropy by adjusting the neuron's weights, we compute the gradient of the conditional entropy with respect to the weights:

$$\frac{\partial h(\Omega_i)}{\partial w_{ij}} = -\int_{\Omega_i} g(\xi)\frac{\partial \log(g(\xi))}{\partial w_{ij}}\big(\log(g(\xi)) + 1\big)d\xi. \quad (6)$$

For a differentiable neuron model, $\partial \log(g(\xi))/\partial w_{ij}$ can be expressed as follows when neuron $i$ fires once at time $\hat{f}_i$ [18]:

$$\frac{\partial \log(g(\xi))}{\partial w_{ij}} = \int_{t=0}^{T} \frac{\partial \rho(u_i(t))}{\partial u_i(t)}\frac{\partial u_i(t)}{\partial w_{ij}}\frac{\left(\delta(t - \hat{f}_i) - \rho(u_i(t))\right)}{\rho(u_i(t))}dt, \quad (7)$$

where $\delta(.)$ is the Dirac delta, and $\rho(u_i(t))$ is the firing probability-density of neuron $i$ at time $t$. (See [18] for the generalization to multiple postsynaptic spikes.) With the sSRM we can compute the partial derivatives $\partial \rho(u_i(t))/\partial u_i(t)$ and $\partial u_i(t)/\partial w_{ij}$. Given the density function (4),

$$\frac{\partial \rho(u_i(t))}{\partial u_i(t)} = \frac{\beta}{1 + \exp(\alpha(\vartheta - u_i(t)))}, \quad \frac{\partial u_i(t)}{\partial w_{ij}} = \varepsilon(t - \hat{f}_i, t - f_j).$$

To perform gradient descent in the conditional entropy, we use the weight update

$$\Delta w_{ij} \propto -\frac{\partial h(\Omega_i)}{\partial w_{ij}} \quad (8)$$

$$\propto \int_{\Omega_i} g(\xi)\big(\log(g(\xi)) + 1\big)\left(\int_{t=0}^{T} \frac{\beta\varepsilon(t - \hat{f}_i, t - f_j)\left(\delta(t - \hat{f}_i) - \rho(u_i(t))\right)}{(1 + \exp(\alpha(\vartheta - u_i(t)))\rho(u_i(t))}dt\right)d\xi.$$

We can use numerical methods to evaluate Equation (8). However, it seems biologically unrealistic to suppose a neuron can integrate over *all* possible responses $\xi$. This dilemma can be circumvented in two ways. First, the resulting learning rule might be cached in some form through evolution so that the full computation is not necessary (e.g., in an STDP curve). Second, the specific response produced by a neuron on a single trial might be considered to be a sample from the distribution $g(\xi)$, and the integration is performed by a sampling process over repeated trials;

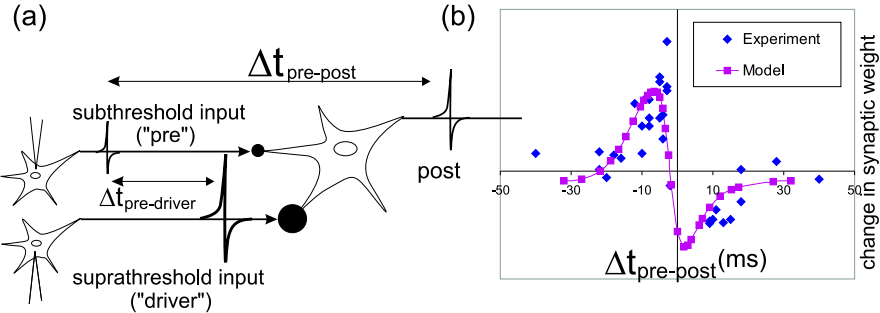

Figure 2: (a) Experimental setup of Zhang *et* al. and (b) their experimental STDP curve (small squares) vs. our model (solid line). Model parameters: $\tau_s = 1.5$ms, $\tau_m = 12.25$ms.

each trial would produce a stochastic gradient step.

## 4 Simulation Methodology

We model in detail the experiment of Zhang *et al.* [12] (Figure 2a). In this experiment, a *post* neuron is identified that has two neurons projecting to it, call them the *pre* and the *driver*. The pre is subthreshold: it produces depolarization but no spike. The driver is suprathreshold: it induces a spike in the post. Plasticity of the pre-post synapse is measured as a function of the timing between pre and post spikes ($\Delta t_{pre-post}$) by varying the timing between induced spikes in the pre and the driver ($\Delta t_{pre-driver}$). This measurement yields the well-known STDP curve (Figure 1b).[1] The experiment imposes several constraints on a simulation: The driver alone causes spiking > 70% of the time, the pre alone causes spiking < 10% of the time, synchronous firing of driver and pre cause LTP if and only if the post fires, and the time constants of the EPSPs—$\tau_s$ and $\tau_m$ in the sSRM—are in the range of 1–3ms and 10–15ms respectively. These constraints remove many free parameters from our simulation. We do not explicitly model the two input cells; instead, we model the EPSPs they produce. The magnitude of these EPSPs are picked to satisfy the experimental constraints: the driver EPSP alone causes a spike in the post on 77.4% of trials, and the pre EPSP alone causes a spike on fewer than 0.1% of trials. Free parameters of the simulation are $\vartheta$ and $\beta$ in the spike-probability function ($\alpha$ can be folded into $\vartheta$), and the magnitude ($u_r^s, u_{abs}$) and reset time constants ($\tau_r^s, \tau_r^f, \Delta_{abs}$).

The dependent variable of the simulation is $\Delta t_{pre-driver}$, and we measure the time of the post spike to determine $\Delta t_{pre-post}$. We estimate the weight update for a given $\Delta t_{pre-driver}$ using Equation 8, approximating the integral by a summation over all time-discretized output responses consisting of 0, 1, or 2 spikes. Three or more spikes have a probability that is vanishingly small.

## 5 Results

Figure 2b shows a typical STDP curve obtained from the model by plotting the estimated weight update of Equation 8 against $\Delta t_{pre-post}$. The model also explains a key finding that has not been explained by any other account, namely, that the magnitude of LTP or LTD decreases as the efficacy of the synapse between the pre and the post increases [2]. Further, the dependence is stronger for LTP than LTD. Figure 3a plots the magnitude of LTP for $\Delta t_{pre-post} = -5$ ms and the magnitude of LTD for $\Delta t_{pre-post} = 7$ ms as the amplitude of the pre's EPSP is increased. The magnitude of the weight change decreases as the weight increases, and this

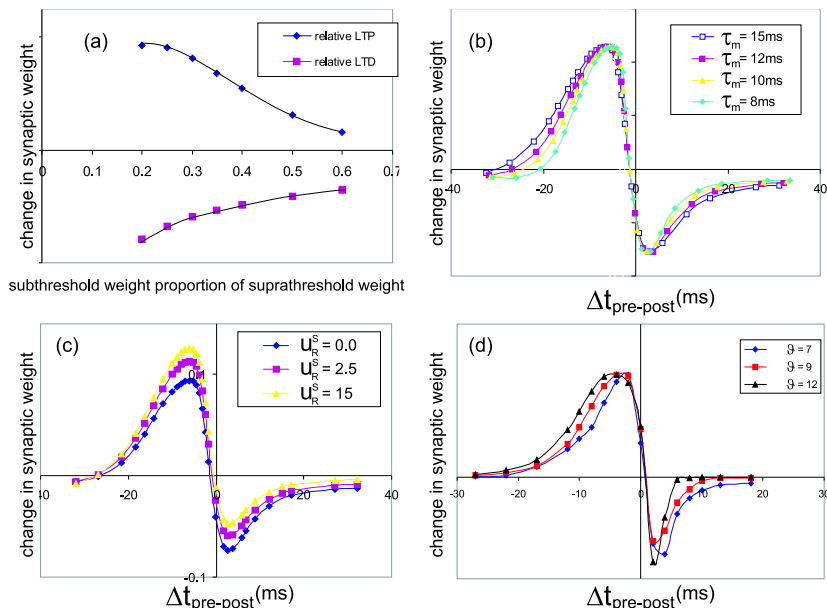

Figure 3: (a) LTP and LTD plasticity as a function of synaptic efficacy of the subthreshold input. (b)-(d) STDP curves predicted by model as $\tau_m$, $u_r^s$, and $\vartheta$ are manipulated.

effect is stronger for LTP than LTD. The model's explanation for this phenomenon is simple: As the weight increases, its effect saturates, and a small change to the weight does little to alter its influence. Consequently, the gradient of the entropy with respect to the weight goes toward zero.

The qualitative shape of the STDP curve is robust to settings of the model's parameters, e.g., the EPSP decay time constant $\tau_m$ (Figure 3b), the strength of the threshold reset $u_r^s$ (Figure 3c), and the spiking threshold $\vartheta$ (Figure 3d). Additionally, the spike-probability function (exponential, sigmoidal, or linear) is not critical. The model makes two predictions relating the shape of the STDP curve to properties of a neuron. These predictions are empirically testable if a diverse population of cells can be studied: (1) the width of the LTD and LTP windows should depend on the EPSP decay time constant (Figure 3b), (2) the strength of LTP to LTD should depend on the strength of the threshold reset (Figure 3c), because stronger resets lead to reduced LTD by reducing the probability of a second spike.

# 6 Discussion

In this paper, we explored a fundamental computational principle, that synapses adapt so as to minimize the variability of a neuron's response in the face of noisy inputs, yielding more reliable neural representations. From this principle—instantiated as conditional entropy minimization—we derived the STDP learning curve. Importantly, the simulation methodology we used to derive the curve closely follows the procedure used in neurophysiological experiments [12]. Our simulations obtain an STDP curve that is robust to model parameters and details of the noise distribution.

Our results are critically dependent on the use of Gerstner's stochastic Spike Response Model, whose dynamics are a good approximation to those of a biological spiking neuron. The sSRM has the virtue of being characterized by parameters that are readily related to neural dynamics, and its dynamics are differentiable, allowing us to derive a gradient-descent learning rule.

Our simulations are based on the classical STDP experiment in which a single presynaptic spike is paired with a single postsynaptic spike. The same methodology can be applied to the situation in which there are multiple presynaptic and/or postsynaptic spikes, although the computation involved becomes nontrivial. We are currently modeling the data from multi-spike experiments.

We modeled the Zhang *et al.* experiment in which a driver neuron is used to induce the post to fire. To induce the post to fire, most other studies use a depolarizing current injection. We are not aware of any established model for current injection within the SRM framework, and we are currently elaborating such a model. We expect to then be able to simulate experiments in which current injections are used, allowing us to investigate the interesting issue of whether the two experimental techniques produce different forms of STDP.

**Acknowledgement**  Work of SMB supported by the Netherlands Organization for Scientific Research (NWO), TALENT grant S-62 588.

## Footnotes

[1]In most experimental studies of STDP, the driver neuron is not used: the post is induced to spike by a direct depolarizing current injection. Modeling current injections requires additional assumptions. Consequently, we focus on the Zhang *et al.* experiment.

# References

[1] G-q. Bi and M-m. Poo. Synaptic modification by correlated activity: Hebb's postulate revisited. *Ann. Rev. Neurosci.*, 24:139–166, 2001.

[2] A. Kepecs, M.C.W. van Rossum, S. Song, and J. Tegner. Spike-timing-dependent plasticity: common themes and divergent vistas. *Biol. Cybern.*, 87:446–458, 2002.

[3] A. Saudargiene, B. Porr, and F. Wörgötter. How the shape of pre- and postsynaptic signals can influence stdp: A biophysical model. *Neural Comp.*, 16:595–625, 2004.

[4] W. Gerstner, R. Kempter, J. L. van Hemmen, and H. Wagner. A neural learning rule for sub-millisecond temporal coding. *Nature*, 383:76–78, 1996.

[5] S. Song, K. Miller, and L. Abbott. Competitive hebbian learning through spiketime-dependent synaptic plasticity. *Nat. Neurosci.*, 3:919–926, 2000.

[6] R. van Rossum, G.-q. Bi, and G.G. Turrigiano. Stable hebbian learning from spike time dependent plasticity. *J. Neurosci.*, 20:8812–8821, 2000.

[7] L.F. Abbott and W. Gerstner. Homeostasis and Learning through STDP. In D. Hansel *et al*(eds), *Methods and Models in Neurophysics*, 2004.

[8] R.P.N. Rao and T.J. Sejnowski. Spike-timing-dependent plasticity as temporal difference learning. *Neural Comp.*, 13:2221–2237, 2001.

[9] P. Dayan and M. Häusser. Plasticity kernels and temporal statistics. In S. Thrun, L. Saul, and B. Schölkopf, editors, *NIPS 16*. 2004.

[10] G. Chechik. Spike-timing-dependent plascticity and relevant mutual information maximization. *Neural Comp.*, 15:1481–1510, 2003.

[11] R.C. Froemke and Y. Dan. Spike-timing-dependent synaptic modification induced by natural spike trains. *Nature*, 416:433–438, 2002.

[12] L.l. Zhang, H.W. Tao, C.E. Holt, W.A. Harris, and M-m. Poo. A critical window for cooperation and competition among developing retinotectal synapses. *Nature*, 395:37–44, 1998.

[13] W. Gerstner. A framework for spiking neuron models: The spike response model. In F. Moss & S. Gielen (eds), *The Handbook of Biol. Physics*, vol 4, pp 469–516, 2001.

[14] A.J. Bell and L.C. Parra. Maximizing information yields spike timing dependent plasticity. *NIPS 17*. 2005.

[15] T. Toyoizumi, J-P. Pfister, K. Aihara, and W. Gerstner. Spike-timing dependent plasticity and mutual information maximization for a spiking neuron model. *NIPS 17*. 2005.

[16] R. Jolivet, T.J. Lewis, and W. Gerstner. The spike response model: a framework to predict neuronal spike trains. In Kaynak *et al.*(eds), *Proc. ICANN/ICONIP 2003*, pp 846–853. 2003.

[17] A. Herrmann and W. Gerstner. Noise and the PSTH response to current transients: I. *J. Comp. Neurosci.*, 11:135–151, 2001.

[18] X. Xie and H.S. Seung. Learning in neural networks by reinforcement of irregular spiking. *Physical Review E*, 69(041909), 2004.
